# Reinforcement Learning in Robust Markov Decision Processes

**Shiau Hong Lim**
Department of Mechanical Engineering
National University of Singapore
Singapore
mpelsh@nus.edu.sg

**Huan Xu**
Department of Mechanical Engineering
National University of Singapore
Singapore
mpexuh@nus.edu.sg

**Shie Mannor**
Department of Electrical Engineering
Technion, Israel
shie@ee.technion.ac.il

## Abstract

An important challenge in Markov decision processes is to ensure robustness with respect to unexpected or adversarial system behavior while taking advantage of well-behaving parts of the system. We consider a problem setting where some unknown parts of the state space can have arbitrary transitions while other parts are purely stochastic. We devise an algorithm that is adaptive to potentially adversarial behavior and show that it achieves similar regret bounds as the purely stochastic case.

## 1 Introduction

Markov decision processes (MDPs) [Puterman, 1994] have been widely used to model and solve sequential decision problems in stochastic environments. Given the parameters of an MDP, namely, the rewards and transition probabilities, an optimal policy can be computed. In practice, these parameters are often estimated from noisy data and furthermore, they may change during the execution of a policy. Hence, the performance of the chosen policy may deteriorate significantly; see [Mannor et al., 2007] for numerical experiments.

The robust MDP framework has been proposed to address this issue of parameter uncertainty (e.g., [Nilim and El Ghaoui, 2005] and [Iyengar, 2005]). The robust MDP setting assumes that the true parameters fall within some uncertainty set $\mathcal{U}$ and seeks a policy that performs the best under the worst realization of the parameters. These solutions, however, can be overly conservative since they are based on worst-case realization. Variants of robust MDP formulations have been proposed to mitigate the conservativeness when additional information on parameter distribution [Strens, 2000, Xu and Mannor, 2012] or coupling among the parameters [Mannor et al., 2012] are known. A major drawback of previous work on robust MDPs is that they all focused on the *planning problem* with no effort to learn the uncertainty. Since in practice it is often difficult to accurately quantify the uncertainty, the solutions to the robust MDP can be conservative if a too large uncertainty set is used.

In this work, we make the first attempt to perform *learning* in robust MDPs. We assume that some of the state-action pairs are adversarial in the sense that their parameters can change arbitrarily within $\mathcal{U}$ from one step to another. However, others are benign in the sense that they are fixed and behave purely stochastically. The learner, however, is given only the uncertainty set $\mathcal{U}$ and knows neither the parameters nor the true nature of each state-action pair.

In this setting, a traditional robust MDP approach would be equivalent to assuming that all parameters are adversarial and therefore would always execute the minimax policy. This is too conservative since it could be the case that most of the parameters are stochastic. Alternatively, one could use an existing online learning algorithm such as UCRL2 [Jaksch et al., 2010] and assume that all parameters are stochastic. This, as we show in the next section, may lead to suboptimal performance when some of the states are adversarial.

Instead, we propose an online learning approach to robust MDPs. We show that the cumulative reward obtained from this method is as good as the minimax policy *that knows the true nature of each state-action pair*. This means that by incorporating learning in robust MDPs, we can effectively resolve the "conservativeness due to not knowing the uncertainty" effect.

The rest of the paper is structured as follows. Section 2 discusses the key difficulties in our setting and explains why existing solutions are not applicable. In subsequent sections, we present our algorithm, its theoretical performance bound and its analysis. Sections 3 and 4 cover the finite-horizon case while Section 5 deals with the infinite-horizon case. We present some experiment results in Section 6 and conclude in Section 7.

## 2   Problem setting

We consider an MDP $M$ with a finite state space $\mathcal{S}$ and a finite action space $\mathcal{A}$. Let $S = |\mathcal{S}|$ and $A = |\mathcal{A}|$. Executing action $a$ in state $s$ results in a random transition according to a distribution $p_{s,a}(\cdot)$ where $p_{s,a}(s')$ gives the probability of transitioning to state $s'$, and accumulate an immediate reward $r(s,a)$.

A robust MDP considers the case where the transition probability is determined in an adversarial way. That is, when action $a$ is taken at state $s$, the transition probability $p_{s,a}(\cdot)$ can be an arbitrary element of the *uncertainty set* $\mathcal{U}(s,a)$. In particular, for different visits of same $(s,a)$, the realization of $p_{s,a}$ can be different, possibly depends on the history. This can model cases where the system dynamics are influenced by competitors or exogenous factors that are hard to model, or the MDP is a simplification of a complicated dynamic system.

Previous research in robust MDPs focused exclusively on the *planning problem*. Here, the power of the adversary – the uncertainty set of the parameter – is precisely known, and the goal is to find the minimax policy – the policy with the best performance under the worst admissible parameters.

This paper considers the *learning problem* of robust MDPs. We ask the following question: suppose the power of the adversary (the extent to which it can affect the system) is not completely revealed to the decision maker, if we are allowed to play the MDP many times, can we still obtain an optimal policy as if we knew the true extent of its power? Or to put it that way, can we develop a procedure that provides the *exact amount of protection* against the *unknown* adversary?

Our specific setup is as follows: for each $(s,a) \in \mathcal{S} \times \mathcal{A}$ an uncertainty set $\mathcal{U}(s,a)$ is given. However, *not all states are adversarial*. Only a subset $\mathcal{F} \subset \mathcal{S} \times \mathcal{A}$ is truly adversarial while all the other state-action pairs behave purely stochastically, i.e., with a fixed unknown $p_{s,a}$. Moreover, the set $\mathcal{F}$ is *not known to the algorithm*.

This setting differs from existing setups, and is challenging for the following reasons:

1. The adversarial actions $p_{s,a}$ are not directly observable.
2. The adversarial behavior is not constrained, except it must belong to the uncertainty set.
3. Ignoring the adversarial component results in sub-optimal behavior.

The first challenge precludes the use of algorithm based on stochastic games such as R-Max [Brafman and Tennenholtz, 2002]. The R-Max algorithm deals with stochastic games where the opponent's action-set for each state is known and the opponent's actions are always observable. In our setting, only the outcome (i.e., the next-state and the reward) of each transition is observable. The algorithm does not observe the action $p_{s,a}$ taken by the adversary. Indeed, because the set $\mathcal{F}$ is unknown, even the action set of the adversary is unknown to the algorithm.

The second challenge is due to unconstrained adversarial behavior. For state-action pairs $(s,a) \in \mathcal{F}$, the opponent is free to choose any $p_{s,a} \in \mathcal{U}(s,a)$ for each transition, possibly depends on the his-

tory and the strategy of the decision maker (i.e., non-oblivious). This affects the sort of performance guarantee one can reasonably expect from any algorithms. In particular, when considering the regret against the best stationary policy "in hindsight", [Yu and Mannor, 2009] show that small change in *transition probabilities* can cause large regret. Even with additional constraints on the allowed adversarial behavior, they showed that the regret bound still does not vanish with respect to the number of steps. Indeed, most results for adversarial MDPs [Even-Dar et al., 2005, Even-Dar et al., 2009, Yu et al., 2009, Neu et al., 2010, Neu et al., 2012] only deal with adversarial rewards while the transitions are assumed stochastic and fixed, which is considerably simpler than our setting.

Since it is not possible to achieve vanishing regret against best stationary policy in hindsight, we choose to measure the regret against the performance of a minimax policy that knows exactly which state-actions are adversarial (i.e., the set $\mathcal{F}$) as well as the true $p_{s,a}$ for all stochastic state-action pairs. Intuitively, this means that if the adversary chooses to play "nicely", we are not constrained to exploit this.

Finally, given that we are competing against the minimax policy, one might ask whether we could simply apply existing algorithms such as UCRL2 [Jaksch et al., 2010] and treat every state-action pair as stochastic. The following example shows that ignoring any adversarial behavior may lead to large regret compared to the minimax policy.

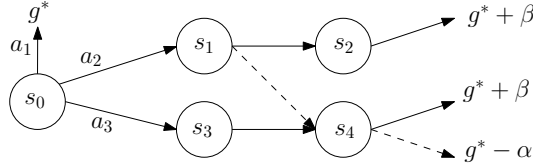

Figure 1: Example MDP with adversarial transitions.

Consider the MDP in Figure 1. Suppose that a UCRL2-like algorithm is used, where all transitions are assumed purely stochastic. There are 3 alternative policies, each corresponds to choosing action $a_1$, $a_2$ and $a_3$ respectively in state $s_0$. Action $a_1$ leads to the optimal minimax average reward of $g^*$. State $s_2$ leads to average reward of $g^* + \beta$ for some $\beta > 0$. State $s_1$ has adversarial transition, where both $s_2$ and $s_4$ are possible next states. $s_4$ has a similar behavior, where it may either lead to $g^* + \beta$ or a "bad" region with average reward $g^* - \alpha$ for some $2\beta < \alpha < 3\beta$.

We consider two phases. In phase 1, the adversary behaves "benignly" by choosing all solid-line transitions. Since both $a_2$ and $a_3$ lead to similar outcome, we assume that in phase 1, both $a_2$ and $a_3$ are chosen for $T$ steps each. In phase 2, the adversary chooses the dashed-line transitions in both $s_1$ and $s_4$. Due to $a_2$ and $a_3$ having similar values (both $g^* + \beta > g^*$) we can assume that $a_2$ is always chosen in phase 2 (if $a_3$ is ever chosen in phase 2 its value will quickly drop below that of $a_2$). Suppose that $a_2$ also runs for $T$ steps in phase 2. A little algebra (see the supplementary material for details) shows that at the end of phase 2 the expected value of $s_4$ (from the learner's point of view) is $g_4 = g^* + \frac{\beta - \alpha}{2}$ and therefore the expected value of $s_1$ is $g_1 = g^* + \frac{3\beta - \alpha}{4} > g^*$. The total accumulated rewards over both phases is however $3Tg^* + T(2\beta - \alpha)$. Let $c = \alpha - 2\beta > 0$. This means that the overall total regret is $cT$ which is linear in $T$.

Note that in the above example, the expected value of $a_2$ remains greater than the minimax value $g^*$ throughout phase 2 and therefore the algorithm will continue to prefer $a_2$, even though the actual accumulated average value is already way below $g^*$. The reason behind this is that the Markov property, which is crucial for UCRL2-like algorithms to work, has been violated due to $s_1$ and $s_4$ behaving in a non-independent way caused by the adversary.

## 3 Algorithm and main result

In this section, we present our algorithm and the main result for the finite-horizon case with the total reward as the performance measure. Section 5 provides the corresponding algorithm and result for the infinite-horizon average-reward case.

For simplicity, we assume without loss of generality a deterministic and known reward function $r(s, a)$. We also assume that rewards are bounded such that $r(s, a) \in [0, 1]$. It is straight-forward, by introducing additional states, to extend the algorithm and analysis to the case where the reward function is random, unknown and even adversarial.

In the finite horizon case, we consider an episodic setting where each episode has a fixed and known length $T$. The algorithm starts at a (possibly random) state $s_0$ and executes $T$ stages. After that, a new episode begins, with an arbitrarily chosen start state (it can simply be the last state of the previous episode). This goes on indefinitely.

Let $\pi$ be a finite-horizon (non-stationary) policy where $\pi_t(s)$ gives the action to be executed in state $s$ at step $t$ in an episode, where $t = 0, \ldots, (T-1)$. Let $P_t$ be a particular choice of $p_{s,a} \in \mathcal{U}(s, a)$ for every $(s, a) \in \mathcal{F}$ at step $t$. For each $t = 0, \ldots, (T-1)$, we define

$$V_t^\pi(s) = \min_{P_t, \ldots, P_{T-2}} \mathbb{E}_{P_t, \ldots, P_{T-2}} \sum_{t'=t}^{T-1} r(s_{t'}, \pi_{t'}(s_{t'})) \qquad \text{and} \qquad V_t^*(s) = \max_\pi V_t^\pi(s),$$

where $s_t = s$ and $s_{t+1}, \ldots, s_{T-1}$ are random variables due to the random transitions. We assume that $\mathcal{U}$ is such that the minimum above exists (e.g., compact set). It is not hard to show that given state $s$, there exists a policy $\pi$ with $V_0^\pi(s) = V_0^*(s)$ and we can compute such a minimax policy if the algorithm knows $\mathcal{F}$ and $p_{s,a}$ for all $(s, a) \notin \mathcal{F}$, from literature of robust MDP (e.g., [Nilim and El Ghaoui, 2005] and [Iyengar, 2005]).

The main message of this paper is that we can determine a policy as good as the minimax policy *without knowing either $\mathcal{F}$ or $p_{s,a}$ for $(s, a) \notin \mathcal{F}$.* To make this formal, we define the regret (against the minimax performance) in episode $i$, for $i = 1, 2, \ldots$ as

$$\Delta_i = V_0^*(s_0^i) - \sum_{t=0}^{T-1} r(s_t^i, a_t^i),$$

where $s_t^i$ and $a_t^i$ denote the actual state visited and action taken at step $t$ of episode $i$.[1] The total regret for $m$ episodes, which we want to minimize, is thus defined as

$$\Delta(m) = \sum_{i=1}^{m} \Delta_i.$$

The main algorithm is given in Figure 2. OLRM is basically UCRL2 [Jaksch et al., 2010] with an additional stochastic check to detect adversarial state-action pairs. Like UCRL2, the algorithm employs the "optimism under uncertainty" principle. We start by assuming that all states are stochastic. If the adversary plays "nicely", nothing else would have to be done. The key challenge, however, is to successfully identify the adversarial state-action pairs when they start to behave maliciously.

A similar scenario in the multi-armed bandit setting has been addressed by [Bubeck and Slivkins, 2012]. They show that it is possible to achieve near-optimal regret without knowing a priori whether a bandit is stochastic or adversarial. In [Bubeck and Slivkins, 2012], the key is to check some consistency conditions that would be satisfied if the behavior is stochastic. We use the same strategy and the question is then, which condition? We discuss this in section 3.2.

Note that the index $k = 1, 2, \ldots$ tracks the number of policies. A policy is executed until either a new pair $(s, a)$ fails the stochastic check, and hence deemed to be adversarial, or some state-action pair has been executed too many times. In either case, we need to re-compute the current optimistic policy (see Section 3.1 for the detail). Every time a new policy is computed we call it a new *epoch*. While each episode has the same length ($T$), each epoch can span multiple episodes, and an epoch can begin in the middle of an episode.

## 3.1 Computing an optimistic policy

Figure 3 shows the algorithm for computing the optimistic minimax policy, where we treat all state-action pairs in the set $F$ as adversarial, and (similar to UCRL2) use optimistic values for other state-action pairs.

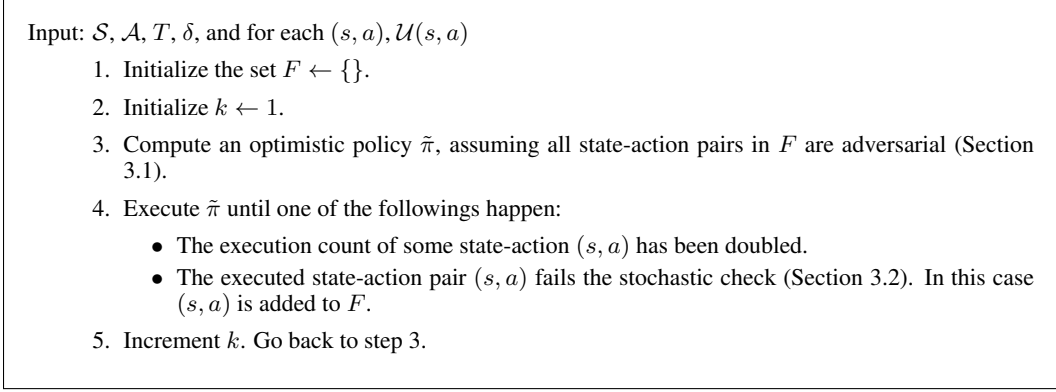

Input: $\mathcal{S}$, $\mathcal{A}$, $T$, $\delta$, and for each $(s,a)$, $\mathcal{U}(s,a)$
1. Initialize the set $F \leftarrow \{\}$.
2. Initialize $k \leftarrow 1$.
3. Compute an optimistic policy $\tilde{\pi}$, assuming all state-action pairs in $F$ are adversarial (Section 3.1).
4. Execute $\tilde{\pi}$ until one of the followings happen:
   - The execution count of some state-action $(s,a)$ has been doubled.
   - The executed state-action pair $(s,a)$ fails the stochastic check (Section 3.2). In this case $(s,a)$ is added to $F$.
5. Increment $k$. Go back to step 3.

Figure 2: The OLRM algorithm

Here, to simplify notations, we frequently use $V(\cdot)$ to mean the vector whose elements are $V(s)$ for each $s \in \mathcal{S}$. This applies to both value functions as well as probability distributions over $\mathcal{S}$. In particular, we use $p(\cdot)V(\cdot)$ to mean the dot product between two such vectors, i.e. $\sum_s p(s)V(s)$. We use $N_k(s,a)$ to denote the total number of times the state-action pair $(s,a)$ has been executed before epoch $k$. The corresponding empirical next-state distribution based on these transitions is denoted as $\hat{P}_k(\cdot|s,a)$. If $(s,a)$ has never been executed before epoch $k$, we define $N_k(s,a) = 1$ and assume $\hat{P}_k(\cdot|s,a)$ to be arbitrarily defined.

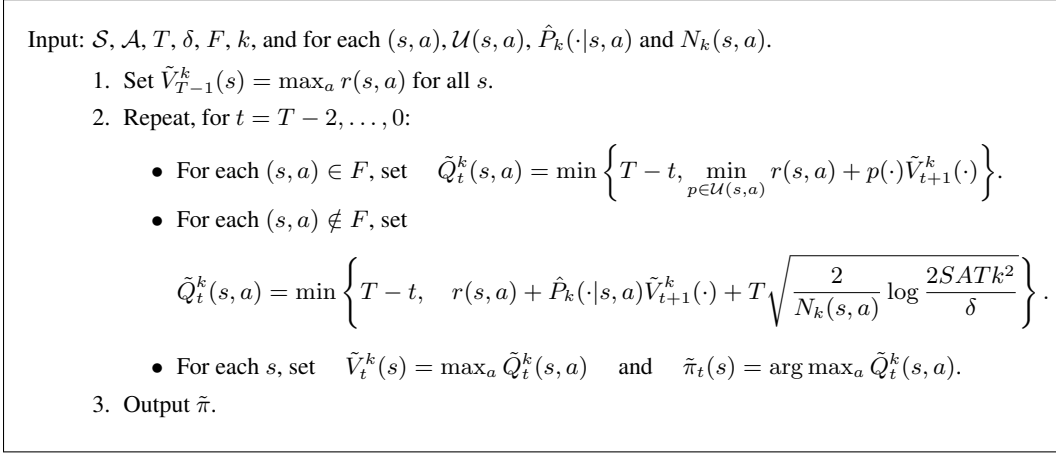

Input: $\mathcal{S}$, $\mathcal{A}$, $T$, $\delta$, $F$, $k$, and for each $(s,a)$, $\mathcal{U}(s,a)$, $\hat{P}_k(\cdot|s,a)$ and $N_k(s,a)$.
1. Set $\tilde{V}_{T-1}^k(s) = \max_a r(s,a)$ for all $s$.
2. Repeat, for $t = T-2, \dots, 0$:
   - For each $(s,a) \in F$, set $\quad \tilde{Q}_t^k(s,a) = \min \left\{ T - t, \min_{p \in \mathcal{U}(s,a)} r(s,a) + p(\cdot)\tilde{V}_{t+1}^k(\cdot) \right\}$.
   - For each $(s,a) \notin F$, set

   $$\tilde{Q}_t^k(s,a) = \min \left\{ T - t, \quad r(s,a) + \hat{P}_k(\cdot|s,a)\tilde{V}_{t+1}^k(\cdot) + T\sqrt{\frac{2}{N_k(s,a)}\log\frac{2SATk^2}{\delta}} \right\}.$$

   - For each $s$, set $\quad \tilde{V}_t^k(s) = \max_a \tilde{Q}_t^k(s,a) \quad$ and $\quad \tilde{\pi}_t(s) = \arg\max_a \tilde{Q}_t^k(s,a)$.
3. Output $\tilde{\pi}$.

Figure 3: Algorithm for computing an optimistic minimax policy.

## 3.2 Stochasticity check

Every time a state-action $(s,a) \notin F$ is executed, the outcome is recorded and subjected to a "stochasticity check". Let $n$ be the total number of times $(s,a)$ has been executed (including the latest one) and $s_1', \dots, s_n'$ are the next-states for each of these transitions. Let $k_1, \dots, k_n$ be the epochs in which each of these transitions happens. Let $t_1, \dots, t_n$ be the step within the episodes (i.e. episode stage) where these transitions happen. Let $\tau$ be the total number of steps executed by the algorithm (from the beginning) so far. The stochastic check fails if:

$$\sum_{j=1}^n \hat{P}_{k_j}(\cdot|s,a)\tilde{V}_{t_j+1}^{k_j}(\cdot) - \sum_{j=1}^n \tilde{V}_{t_j+1}^{k_j}(s_j') > 5T\sqrt{nS\log\frac{4SAT\tau^2}{\delta}}.$$

The stochastic check follows the intuitive saying "if it is not broke, don't fix it", by checking whether the value of actual transition from $(s,a)$ is below what is expected from the parameter estimation.

One can show that with high probability, *all* stochastic state-action pairs will *always* pass the stochastic check. Now consider an adversarial $(s, a)$ pair: if the adversary plays "nicely", the current policy accumulates satisfactory reward and hence nothing needs to be changed, even if the transitions themselves fail to "look" stochastic; if the adversary plays "nasty", then the stochastic check will detect it, and subsequently protect against it.

### 3.3 Main result

The following theorem summarizes the performance of OLRM. Here and in the sequel, we use $\tilde{\mathcal{O}}$ when the log terms are omitted. Our result for the infinite-horizon case is similar (see Section 5).

**Theorem 1.** *Given $\delta$, $T$, $\mathcal{S}$, $\mathcal{A}$, the total regret of OLRM is*

$$\Delta(m) \leq \tilde{\mathcal{O}}(ST^{3/2}\sqrt{Am})$$

*for all $m$, with probability at least $1 - \delta$.*

Note that the above is with respect to the total number of episodes $m$. Since the total number of steps is $\tau = mT$, the regret bound in terms of $\tau$ is therefore $\tilde{\mathcal{O}}(ST\sqrt{A\tau})$. This gives the familiar $\sqrt{\tau}$ regret as in UCRL2. Also, the bound has the same dependencies on $S$ and $A$ as in UCRL2. The horizon length $T$ plays the role of the "diameter" in the infinite-horizon case and again it has the same dependency as its counterpart in UCRL2.

The result shows that even though the algorithm deals with unknown stochastic and potentially adversarial states, it achieves the same regret bound as in the fully stochastic case. In the case where all states are in fact stochastic, this reduces to the same UCRL2 result.

## 4 Analysis of OLRM

We briefly explain the roadmap of the proof of Theorem 1. The complete proof can be found in the supplementary material.

Our proof starts with the following technical Lemma.

**Lemma 1.** *The following holds for all state-action pair $(s, a) \notin \mathcal{F}$ and for $t = 0, \ldots, (T - 1)$ in all epochs $k \geq 1$, with probability at least $1 - \delta$:*

$$\hat{P}_k(\cdot|s,a)\tilde{V}_{t+1}^k(\cdot) - p_{s,a}(\cdot)\tilde{V}_{t+1}^k(\cdot) \leq T\sqrt{\frac{2S}{N_k(s,a)}\log\frac{4SATk^2}{\delta}}.$$

*Proof sketch.* Since $(s, a) \notin \mathcal{F}$ is stochastic, we apply the bound from [Weissman et al., 2003] for the 1-norm deviation between $\hat{P}_k(\cdot|s, a)$ and $p_{s,a}$. The bound follows from $\|\tilde{V}_{t+1}^k(\cdot)\|_\infty \leq T$. $\square$

Using Lemma 1, we show the following lemma that with high probability, all purely stochastic state-action pairs will always pass the stochastic check.

**Lemma 2.** *The probability that any state-action pair $(s, a) \notin \mathcal{F}$ gets added into set $F$ while running the algorithm is at most $2\delta$.*

*Proof sketch.* Each $(s, a) \notin \mathcal{F}$ is purely stochastic. Suppose $(s, a)$ has been executed $n$ times and $s'_1, \ldots, s'_n$ are the next-states for these transitions. Recall that the check fails if

$$\sum_{j=1}^n \hat{P}_{k_j}(\cdot|s,a)\tilde{V}_{t_j+1}^{k_j}(\cdot) - \sum_{j=1}^n \tilde{V}_{t_j+1}^{k_j}(s'_j) > 5T\sqrt{nS\log\frac{4SAT\tau^2}{\delta}}.$$

We can derive a high-probability bound that satisfies the stochastic check by applying the Azuma-Hoeffding inequality on the martingale difference sequence

$$X_j = p_{s,a}(\cdot)\tilde{V}_{t_j+1}^{k_j}(\cdot) - \tilde{V}_{t_j+1}^{k_j}(s'_j)$$

followed by an application of Lemma 1. $\square$

We then show that all value estimates $\tilde{V}_t^k$ are always optimistic.

**Lemma 3.** *With probability at least $1 - \delta$, and assume that no state-action pairs $(s, a) \notin \mathcal{F}$ have been added to $F$, the following holds for every state $s \in \mathcal{S}$, every $t \in \{0, \ldots, T-1\}$ and every $k \geq 1$:*

$$\tilde{V}_t^k(s) \geq V_t^*(s).$$

*Proof sketch.* The key challenge is to prove that state-actions in $\mathcal{F}$ (adversarial) that have not been identified (i.e. all past transitions passed the test) would have optimistic $\tilde{Q}$ values. This can be done by, again, applying the Azuma-Hoeffding inequality. $\qquad\square$

Equipped with the previous three lemmas, we are now able to establish Theorem 1.

*Proof sketch.* Lemma 3 established that all value estimates $\tilde{V}_t^k$ are always optimistic. We can therefore bound the regret by bounding the difference between $\tilde{V}_t^k$ and the actual rewards received by the algorithm. The "optimistic gap" shrinks in an expected manner as the number of steps executed by the algorithm grows if all state-actions are stochastic.

For an adversarial state-action $(s, a) \in \mathcal{F}$, we use the following facts to ensure the above: (i) If $(s, a)$ has been added to $F$ (i.e., it failed the stochastic check) then all policies afterwards would correctly evaluate its value; (ii) All transitions before $(s, a)$ is added to $F$ (if ever) must have passed the stochastic check and the check condition ensures that its behavior is consistent with what one would expect if $(s, a)$ was stochastic. $\qquad\square$

## 5 Infinite horizon case

In the infinite horizon case, let $P$ be a particular choice of $p_{s,a} \in \mathcal{U}(s, a)$ for every $(s, a) \in \mathcal{F}$. Given a (stationary) policy $\pi$, its average undiscounted reward (or "gain") is defined as follows:

$$g_P^\pi(s) = \lim_{\tau \to \infty} \frac{1}{\tau} \mathbb{E}_P \left[ \sum_{t=1}^{\tau} r(s_i, \pi(s_i)) \right]$$

where $s_1 = s$. The limit always exists for finite MDPs [Puterman, 1994]. We make the assumption that regardless of the choice of $P$, the resulting MDP is communicating and unichain. [2] In this case $g_P^\pi(s)$ is a constant and independent of $s$ so we can drop the argument $s$.

We define the worst-case average reward of $\pi$ over all possible $P$ as $g^\pi = \min_P g_P^\pi$. An optimal minimax policy $\pi^*$ is any policy whose gain $g^{\pi^*} = g^* = \max_\pi g^\pi$. We define the regret after executing the MDP $M$ for $\tau$ steps as

$$\Delta(\tau) = \tau g^* - \sum_{t=1}^{\tau} r(s_t, a_t).$$

The main algorithm for the infinite-horizon case, which we refer as OLRM2, is essentially identical to OLRM. The main difference is in computing the optimistic policy and the corresponding stochastic check. The detailed algorithm is presented in the supplementary material.

The algorithms from [Tewari and Bartlett, 2007] can be used to compute an optimistic minimax policy. In particular, for each $(s, a) \in F$, its transition function is chosen pessimistically from $\mathcal{U}(s, a)$. For each $(s, a) \notin F$, its transition function is chosen optimistically from the following set:

$$\{p : \|p(\cdot) - \hat{P}_k(\cdot|s, a)\|_1 \leq \sigma\} \quad \text{where} \quad \sigma = \sqrt{\frac{2S}{N_k(s, a)} \log \frac{4SAk^2}{\delta}}.$$

Let $\tilde{P}_k(\cdot|s, \tilde{\pi}^k(s))$ be the minimax choice of transition functions for each $s$ where the minimax gain $g^{\tilde{\pi}^k}$ is attained. The bias $h_k$ can be obtained by solving the following system of equations for $h(\cdot)$ (see [Puterman, 1994]):

$$\forall s \in \mathcal{S}, \quad g^{\tilde{\pi}^k} + h(s) = r(s, \tilde{\pi}^k(s)) + \tilde{P}_k(\cdot|s, \tilde{\pi}^k(s))h(\cdot). \tag{1}$$

The stochastic check for the infinite-horizon case is mostly identical to the finite-horizon case, except that we replace $T$ with the maximal span $\tilde{H}$ of the bias, defined as follows:

$$\tilde{H} = \max_{k \in \{k_1, \ldots, k_n\}} \left( \max_s h_k(s) - \min_s h_k(s) \right).$$

The stochastic check fails if:

$$\sum_{j=1}^n \tilde{P}_{k_j}(\cdot|s, a)h_{k_j}(\cdot) - \sum_{j=1}^n h_{k_j}(s'_j) > 5\tilde{H}\sqrt{nS \log \frac{4SA\tau^2}{\delta}}.$$

Let $H$ be the maximal span of the bias of any optimal minimax policies. The following summarizes the performance of OLRM2. The proof, deferred in the supplementary material, is similar to Theorem 1.

**Theorem 2.** *Given $\delta$, $\mathcal{S}$, $\mathcal{A}$, the total regret of OLRM2 is*

$$\Delta(\tau) \leq \tilde{\mathcal{O}}(SH\sqrt{A\tau})$$

*for all $\tau$, with probability at least $1 - \delta$.*

## 6  Experiment

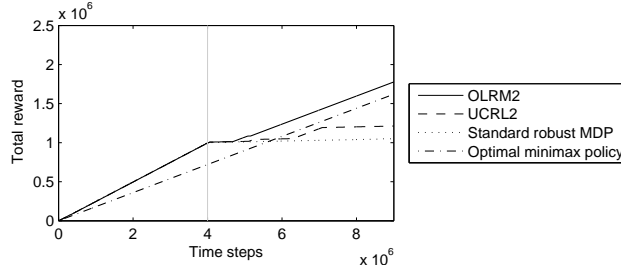

Figure 4: Total accumulated rewards. The vertical line marks the start of "breakdown".

We run both our algorithm as well as UCRL2 on the example MDP in Figure 1 for the infinite-horizon case. Figure 4 shows the result for $g^* = 0.18$, $\beta = 0.07$ and $\alpha = 0.17$. It shows that UCRL2 accumulates smaller total rewards than the optimal minimax policy while our algorithm actually accumulates larger total rewards than the minimax policy. We also include the result for a standard robust MDP that treats all state-action pairs as adversarial and therefore performs poorly. Additional details are provided in the supplementary material.

## 7  Conclusion

We presented an algorithm for online learning of robust MDPs with unknown parameters, some can be adversarial. We show that it achieves similar regret bound as in the fully stochastic case. A natural extension is to allow the learning of the uncertainty sets in adversarial states, where the true uncertainty set is unknown. Our preliminary results show that very similar regret bounds can be obtained for learning from a class of nested uncertainty sets.

### Acknowledgments

This work is partially supported by the Ministry of Education of Singapore through AcRF Tier Two grant R-265-000-443-112 and NUS startup grant R-265-000-384-133. The research leading to these results has received funding from the European Research Council under the European Union's Seventh Framework Programme (FP/2007-2013)/ ERC Grant Agreement n.306638.

## Footnotes

[1] We provide high-probability regret bounds for any single trial, from which the expected regret can be readily derived, if desired.

[2] In more general settings, such as communicating or weakly communicating MDPs, although the optimal policies (for a fixed $P$) always have constant gain, the optimal minimax policies (over all possible $P$) might have non-constant gain. Additional assumptions on $\mathcal{U}$, as well as a slight change in the definition of the regret are needed to deal with these cases. This is left for future research.

# References

[Brafman and Tennenholtz, 2002] Brafman, R. I. and Tennenholtz, M. (2002). R-max - a general polynomial time algorithm for near-optimal reinforcement learning. *Journal of Machine Learning Research*, 3:213–231.

[Bubeck and Slivkins, 2012] Bubeck, S. and Slivkins, A. (2012). The best of both worlds: Stochastic and adversarial bandits. *Journal of Machine Learning Research - Proceedings Track*, 23:42.1–42.23.

[Even-Dar et al., 2005] Even-Dar, E., Kakade, S. M., and Mansour, Y. (2005). Experts in a markov decision process. In Saul, L. K., Weiss, Y., and Bottou, L., editors, *Advances in Neural Information Processing Systems 17*, pages 401–408. MIT Press, Cambridge, MA.

[Even-Dar et al., 2009] Even-Dar, E., Kakade, S. M., and Mansour, Y. (2009). Online markov decision processes. *Math. Oper. Res.*, 34(3):726–736.

[Iyengar, 2005] Iyengar, G. N. (2005). Robust dynamic programming. *Math. Oper. Res.*, 30(2):257–280.

[Jaksch et al., 2010] Jaksch, T., Ortner, R., and Auer, P. (2010). Near-optimal regret bounds for reinforcement learning. *J. Mach. Learn. Res.*, 99:1563–1600.

[Mannor et al., 2012] Mannor, S., Mebel, O., and Xu, H. (2012). Lightning does not strike twice: Robust mdps with coupled uncertainty. In *ICML*.

[Mannor et al., 2007] Mannor, S., Simester, D., Sun, P., and Tsitsiklis, J. N. (2007). Bias and variance approximation in value function estimates. *Manage. Sci.*, 53(2):308–322.

[McDiarmid, 1989] McDiarmid, C. (1989). On the method of bounded differences. In *Surveys in Combinatorics*, number 141 in London Mathematical Society Lecture Note Series, pages 148–188. Cambridge University Press.

[Neu et al., 2012] Neu, G., György, A., and Szepesvári, C. (2012). The adversarial stochastic shortest path problem with unknown transition probabilities. *Journal of Machine Learning Research - Proceedings Track*, 22:805–813.

[Neu et al., 2010] Neu, G., György, A., Szepesvári, C., and Antos, A. (2010). Online markov decision processes under bandit feedback. In *NIPS*, pages 1804–1812.

[Nilim and El Ghaoui, 2005] Nilim, A. and El Ghaoui, L. (2005). Robust control of markov decision processes with uncertain transition matrices. *Oper. Res.*, 53(5):780–798.

[Puterman, 1994] Puterman, M. L. (1994). *Markov Decision Processes: Discrete Stochastic Dynamic Programming*. Wiley-Interscience.

[Strens, 2000] Strens, M. (2000). A bayesian framework for reinforcement learning. In *In Proceedings of the Seventeenth International Conference on Machine Learning*, pages 943–950. ICML.

[Tewari and Bartlett, 2007] Tewari, A. and Bartlett, P. (2007). Bounded parameter markov decision processes with average reward criterion. *Learning Theory*, pages 263–277.

[Weissman et al., 2003] Weissman, T., Ordentlich, E., Seroussi, G., Verdu, S., and Weinberger, M. J. (2003). Inequalities for the l1 deviation of the empirical distribution. Technical report, Information Theory Research Group, HP Laboratories.

[Xu and Mannor, 2012] Xu, H. and Mannor, S. (2012). Distributionally robust markov decision processes. *Math. Oper. Res.*, 37(2):288–300.

[Yu and Mannor, 2009] Yu, J. Y. and Mannor, S. (2009). Arbitrarily modulated markov decision processes. In *CDC*, pages 2946–2953.

[Yu et al., 2009] Yu, J. Y., Mannor, S., and Shimkin, N. (2009). Markov decision processes with arbitrary reward processes. *Math. Oper. Res.*, 34(3):737–757.

